# Hierarchical Topic Models and the Nested Chinese Restaurant Process

**David M. Blei**
blei@cs.berkeley.edu

**Thomas L. Griffiths**
gruffydd@mit.edu

**Michael I. Jordan**
jordan@cs.berkeley.edu

**Joshua B. Tenenbaum**
jbt@mit.edu

University of California, Berkeley
Berkeley, CA 94720

Massachusetts Institute of Technology
Cambridge, MA 02139

## Abstract

We address the problem of learning topic hierarchies from data. The model selection problem in this domain is daunting—which of the large collection of possible trees to use? We take a Bayesian approach, generating an appropriate prior via a distribution on partitions that we refer to as the *nested Chinese restaurant process*. This nonparametric prior allows arbitrarily large branching factors and readily accommodates growing data collections. We build a hierarchical topic model by combining this prior with a likelihood that is based on a hierarchical variant of latent Dirichlet allocation. We illustrate our approach on simulated data and with an application to the modeling of NIPS abstracts.

## 1 Introduction

Complex probabilistic models are increasingly prevalent in domains such as bioinformatics, information retrieval, and vision. These domains create fundamental modeling challenges due to their open-ended nature—data sets often grow over time, and as they grow they bring new entities and new structures to the fore. Current statistical modeling tools often seem too rigid in this regard; in particular, classical model selection techniques based on hypothesis testing are poorly matched to problems in which data can continue to accrue and unbounded sets of often incommensurate structures must be considered at each step.

An important instance of such modeling challenges is provided by the problem of learning a topic hierarchy from data. Given a collection of "documents," each of which contains a set of "words," we wish to discover common usage patterns or "topics" in the documents, and to organize these topics into a hierarchy. (Note that while we use the terminology of document modeling throughout this paper, the methods that we describe are general.) In this paper, we develop efficient statistical methods for constructing such a hierarchy which allow it to grow and change as the data accumulate.

We approach this model selection problem by specifying a generative probabilistic model for hierarchical structures and taking a Bayesian perspective on the problem of learning these structures from data. Thus our hierarchies are random variables; moreover, these random variables are specified procedurally, according to an algorithm that constructs the hierarchy as data are made available. The probabilistic object that underlies this approach

is a distribution on partitions of integers known as the *Chinese restaurant process* [1]. We show how to extend the Chinese restaurant process to a hierarchy of partitions, and show how to use this new process as a representation of prior and posterior distributions for topic hierarchies.

There are several possible approaches to the modeling of topic hierarchies. In our approach, each node in the hierarchy is associated with a topic, where a topic is a distribution across words. A document is generated by choosing a path from the root to a leaf, repeatedly sampling topics along that path, and sampling the words from the selected topics. Thus the organization of topics into a hierarchy aims to capture the breadth of usage of topics across the corpus, reflecting underlying syntactic and semantic notions of generality and specificity. This approach differs from models of topic hierarchies which are built on the premise that the distributions associated with parents and children are similar [2]. We assume no such constraint—for example, the root node may place all of its probability mass on function words, with none of its descendants placing any probability mass on function words. Our model more closely resembles the hierarchical topic model considered in [3], though that work does not address the model selection problem which is our primary focus.

## 2   Chinese restaurant processes

We begin with a brief description of the Chinese restaurant process and subsequently show how this process can be extended to hierarchies.

### 2.1   The Chinese restaurant process

The Chinese restaurant process (CRP) is a distribution on partitions of integers obtained by imagining a process by which $M$ customers sit down in a Chinese restaurant with an infinite number of tables.[1] The basic process is specified as follows. The first customer sits at the first table. The $m$th subsequent customer sits at a table drawn from the following distribution:

$$
\begin{aligned}
p(\text{occupied table } i \mid \text{previous customers}) &= \frac{m_i}{\gamma + m - 1} \\
p(\text{next unoccupied table} \mid \text{previous customers}) &= \frac{\gamma}{\gamma + m - 1}
\end{aligned}
\tag{1}
$$

where $m_i$ is the number of previous customers at table $i$, and $\gamma$ is a parameter. After $M$ customers sit down, the seating plan gives a partition of $M$ items. This distribution gives the same partition structure as draws from a Dirichlet process [4]. However, the CRP also allows several variations on the basic rule in Eq. (1), including a data-dependent choice of $\gamma$ and a more general functional dependence on the current partition [5]. This flexibility will prove useful in our setting.

The CRP has been used to represent uncertainty over the number of components in a mixture model. In a *species sampling mixture* [6], each table in the Chinese restaurant is associated with a draw from $p(\beta \mid \eta)$ where $\beta$ is a mixture component parameter. Each data point is generated by choosing a table $i$ from Eq. (1) and then sampling a value from the distribution parameterized by $\beta_i$ (the parameter associated with that table). Given a data set, the posterior under this model has two components. First, it is a distribution over seating plans; the number of mixture components is determined by the number of tables which the data occupy. Second, given a seating plan, the particular data which are sitting at each table induce a distribution on the associated parameter $\beta$ for that table. The posterior can be estimated using Markov chain Monte Carlo [7]. Applications to various kinds of mixture models have begun to appear in recent years; examples include Gaussian mixture models [8], hidden Markov models [9] and mixtures of experts [10].

## 2.2 Extending the CRP to hierarchies

The CRP is amenable to mixture modeling because we can establish a one-to-one relationship between tables and mixture components and a one-to-many relationship between mixture components and data. In the models that we will consider, however, each data point is associated with multiple mixture components which lie along a path in a hierarchy. We develop a hierarchical version of the CRP to use in specifying a prior for such models.

A *nested Chinese restaurant process* can be defined by imagining the following scenario. Suppose that there are an infinite number of infinite-table Chinese restaurants in a city. One restaurant is determined to be the root restaurant and on each of its infinite tables is a card with the name of another restaurant. On each of the tables in those restaurants are cards that refer to other restaurants, and this structure repeats infinitely. Each restaurant is referred to exactly once; thus, the restaurants in the city are organized into an infinitely-branched tree. Note that each restaurant is associated with a level in this tree (e.g., the root restaurant is at level 1 and the restaurants it refers to are at level 2).

A tourist arrives in the city for a culinary vacation. On the first evening, he enters the root Chinese restaurant and selects a table using Eq. (1). On the second evening, he goes to the restaurant identified on the first night's table and chooses another table, again from Eq. (1). He repeats this process for $L$ days. At the end of the trip, the tourist has sat at $L$ restaurants which constitute a path from the root to a restaurant at the $L$th level in the infinite tree described above. After $M$ tourists take $L$-day vacations, the collection of paths describe a particular $L$-level subtree of the infinite tree (see Figure 1a for an example of such a tree).

This prior can be used to model topic hierarchies. Just as a standard CRP can be used to express uncertainty about a possible number of components, the nested CRP can be used to express uncertainty about possible $L$-level trees.

# 3 A hierarchical topic model

Let us consider a data set composed of a *corpus* of documents. Each *document* is a collection of words, where a *word* is an item in a *vocabulary*. Our basic assumption is that the words in a document are generated according to a mixture model where the mixing proportions are random and document-specific. Consider a multinomial variable $z$, and an associated set of distributions over words $p(w \mid z, \beta)$, where $\beta$ is a parameter. These *topics* (one distribution for each possible value of $z$) are the basic mixture components in our model. The document-specific mixing proportions associated with these components are denoted by a vector $\theta$. Temporarily assuming $K$ possible topics in the corpus, an assumption that we will soon relax, $z$ thus ranges over $K$ possible values and $\theta$ is a $K$-dimensional vector. Our document-specific mixture distribution is $p(w \mid \theta) = \sum_{i=1}^{K} \theta_i p(w \mid z = i, \beta_i)$ which is a random distribution since $\theta$ is random.

We now specify the following two-level generative probabilistic process for generating a document: (1) choose a $K$-vector $\theta$ of topic proportions from a distribution $p(\theta \mid \alpha)$, where $\alpha$ is a corpus-level parameter; (2) repeatedly sample words from the mixture distribution $p(w \mid \theta)$ for the chosen value of $\theta$. When the distribution $p(\theta \mid \alpha)$ is chosen to be a Dirichlet distribution, we obtain the latent Dirichlet allocation model (LDA) [11]. LDA is thus a two-level generative process in which documents are associated with topic proportions, and the corpus is modeled as a Dirichlet distribution on these topic proportions.

We now describe an extension of this model in which the topics lie in a hierarchy. For the moment, suppose we are given an $L$-level tree and each node is associated with a topic. A document is generated as follows: (1) choose a path from the root of the tree to a leaf; (2) draw a vector of topic proportions $\theta$ from an $L$-dimensional Dirichlet; (3) generate the words in the document from a mixture of the topics along the path from root to leaf, with

mixing proportions $\theta$. This model can be viewed as a fully generative version of the cluster abstraction model [3].

Finally, we use the nested CRP to relax the assumption of a fixed tree structure. As we have seen, the nested CRP can be used to place a prior on possible trees. We also place a prior on the topics $\beta_i$, each of which is associated with a restaurant in the infinite tree (in particular, we assume a symmetric Dirichlet with hyperparameter $\eta$). A document is drawn by first choosing an $L$-level path through the restaurants and then drawing the words from the $L$ topics which are associated with the restaurants along that path. Note that all documents share the topic associated with the root restaurant.

1. Let $c_1$ be the root restaurant.
2. For each level $\ell \in \{2, \ldots, L\}$:
   (a) Draw a table from restaurant $c_{\ell-1}$ using Eq. (1). Set $c_\ell$ to be the restaurant referred to by that table.
3. Draw an $L$-dimensional topic proportion vector $\theta$ from $\mathrm{Dir}(\alpha)$.
4. For each word $n \in \{1, \ldots, N\}$:
   (a) Draw $z \in \{1, \ldots, L\}$ from $\mathrm{Mult}(\theta)$.
   (b) Draw $w_n$ from the topic associated with restaurant $c_z$.

This model, *hierarchical LDA* (hLDA), is illustrated in Figure 1b. The node labeled $T$ refers to a collection of an infinite number of $L$-level paths drawn from a nested CRP. Given $T$, the $c_{m,\ell}$ variables are deterministic—simply look up the $\ell$th level of the $m$th path in the infinite collection of paths. However, not having observed $T$, the distribution of $c_{m,\ell}$ will be defined by the nested Chinese restaurant process, conditioned on all the $c_{q,\ell}$ for $q < m$.

Now suppose we are given a corpus of $M$ documents, $\mathbf{w}_1, \ldots, \mathbf{w}_M$. The posterior on the $c$'s is essentially transferred (via the deterministic relationship), to a posterior on the first $M$ paths in $T$. Consider a new document $\mathbf{w}_{M+1}$. Its posterior path will depend, through the unobserved $T$, on the posterior paths of all the documents in the original corpus. Subsequent new documents will also depend on the original corpus and any new documents which were observed before them. Note that, through Eq. (1), any new document can choose a previously unvisited restaurant at any level of the tree. I.e., even if we have a peaked posterior on $T$ which has essentially selected a particular tree, a new document can change that hierarchy if its words provide justification for such a change.

In another variation of this model, we can consider a process that flattens the nested CRP into a standard CRP, but retains the idea that a tourist eats $L$ meals. That is, the tourist eats $L$ times in a single restaurant under the constraint that he does not choose the same table twice. Though the vacation is less interesting, this model provides an interesting prior. In particular, it can be used as a prior for a flat LDA model in which each document can use at most $L$ topics from the potentially infinite total set of topics. We examine such a model in Section 5 to compare CRP methods with selection based on Bayes factors.

## 4   Approximate inference by Gibbs sampling

In this section, we describe a Gibbs sampling algorithm for sampling from the posterior nested CRP and corresponding topics in the hLDA model. The Gibbs sampler provides a method for simultaneously exploring the parameter space (the particular topics of the corpus) and the model space ($L$-level trees).

The variables needed by the sampling algorithm are: $w_{m,n}$, the $n$th word in the $m$th document (the only observed variables in the model); $c_{m,\ell}$, the restaurant corresponding to the $\ell$th topic in document $m$; and $z_{m,n}$, the assignment of the $n$th word in the $m$th document

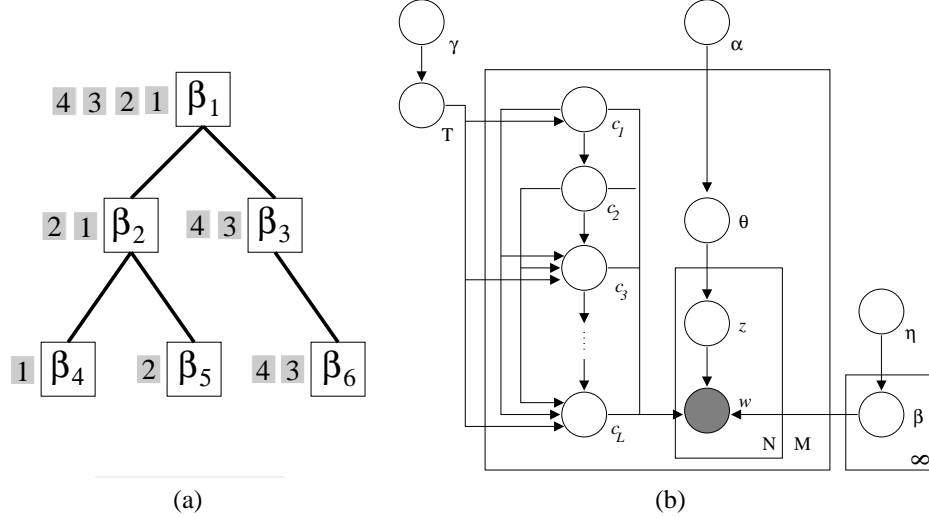

(a)                                   (b)

Figure 1: (a) The paths of four tourists through the infinite tree of Chinese restaurants ($L = 3$). The solid lines connect each restaurant to the restaurants referred to by its tables. The collected paths of the four tourists describe a particular subtree of the underlying infinite tree. This illustrates a sample from the state space of the posterior nested CRP of Figure 1b for four documents. (b) The graphical model representation of hierarchical LDA with a nested CRP prior. We have separated the nested Chinese restaurant process from the topics. Each of the infinite $\beta$'s corresponds to one of the restaurants.

to one of the $L$ available topics. All other variables in the model—$\theta$ and $\beta$—are integrated out. The Gibbs sampler thus assesses the values of $z_{m,n}$ and $c_{m,\ell}$.

Conceptually, we divide the Gibbs sampler into two parts. First, given the current state of the CRP, we sample the $z_{m,n}$ variables of the underlying LDA model following the algorithm developed in [12], which we do not reproduce here. Second, given the values of the LDA hidden variables, we sample the $c_{m,\ell}$ variables which are associated with the CRP prior. The conditional distribution for $\mathbf{c}_m$, the $L$ topics associated with document $m$, is:

$$p(\mathbf{c}_m \mid \mathbf{w}, \mathbf{c}_{-m}, \mathbf{z}) \propto p(\mathbf{w}_m \mid \mathbf{c}, \mathbf{w}_{-m}, \mathbf{z})p(\mathbf{c}_m \mid \mathbf{c}_{-m}),$$

where $\mathbf{w}_{-m}$ and $\mathbf{c}_{-m}$ denote the $\mathbf{w}$ and $\mathbf{c}$ variables for all documents other than $m$. This expression is an instance of Bayes' rule with $p(\mathbf{w}_m \mid \mathbf{c}, \mathbf{w}_{-m}, \mathbf{z})$ as the likelihood of the data given a particular choice of $\mathbf{c}_m$ and $p(\mathbf{c}_m \mid \mathbf{c}_{-m})$ as the prior on $\mathbf{c}_m$ implied by the nested CRP. The likelihood is obtained by integrating over the parameters $\beta$, which gives:

$$p(\mathbf{w}_m \mid \mathbf{c}, \mathbf{w}_{-m}, \mathbf{z}) = \prod_{\ell=1}^{L} \left( \frac{\Gamma(n_{c_{m,\ell},-m}^{(\cdot)} + W\eta)}{\prod_w \Gamma(n_{c_{m,\ell},-m}^{(w)} + \eta)} \frac{\prod_w \Gamma(n_{c_{m,\ell},-m}^{(w)} + n_{c_{m,\ell},m}^{(w)} + \eta)}{\Gamma(n_{c_{m,\ell},-m}^{(\cdot)} + n_{c_{m,\ell},m}^{(\cdot)} + W\eta)} \right),$$

where $n_{c_{m,\ell},-m}^{(w)}$ is the number of instances of word $w$ that have been assigned to the topic indexed by $c_{m,\ell}$, not including those in the current document, $W$ is the total vocabulary size, and $\Gamma(\cdot)$ denotes the standard gamma function. When $\mathbf{c}$ contains a previously unvisited restaurant, $n_{c_{m,\ell},-m}^{(w)}$ is zero.

Note that the $\mathbf{c}_m$ must be drawn as a block. The set of possible values for $\mathbf{c}_m$ corresponds to the union of the set of existing paths through the tree, equal to the number of leaves, with the set of possible novel paths, equal to the number of internal nodes. This set can be enumerated and scored using Eq. (1) and the definition of a nested CRP in Section 2.2.

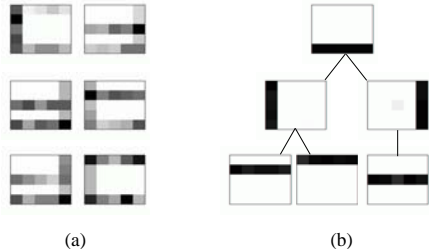

| Structure | Leaf error | | | Other |
|---|---|---|---|---|
| | 0 | 1 | 2 | |
| 3 (7 6 5) | 70% | 14% | 4% | 12% |
| 4 (6 6 5 5) | 48% | 30% | 2% | 20% |
| 4 (6 6 6 4) | 52% | 36% | 0% | 12% |
| 5 (7 6 5 5 4) | 30% | 40% | 16% | 14% |
| 5 (6 5 5 5 4) | 50% | 22% | 16% | 12% |

Figure 2: (a) Six sample documents from a 100 document corpus using the three level bars hierarchy described in Section 5 and $\alpha$ skewed toward higher levels. Each document has 1000 words from a 25 term vocabulary. (b) The correct hierarchy found by the Gibbs sampler on this corpus.

Figure 3: Results of estimating hierarchies on simulated data. *Structure* refers to a three level hierarchy: the first integer is the number of branches from the root and is followed by the number of children of each branch. *Leaf error* refers to how many leaves were incorrect in the resulting tree (0 is exact). *Other* subsumes all other errors.

## 5   Examples and empirical results

In this section we describe a number of experiments using the models described above. In all experiments, we let the sampler burn in for 10000 iterations and subsequently took samples 100 iterations apart for another 1000 iterations. Local maxima can be a problem in the hLDA model. To avoid them, we randomly restart the sampler 25 times and take the trajectory with the highest average posterior likelihood.

We illustrate that the nested CRP process is feasible for learning text hierarchies in hLDA by using a contrived corpus on a small vocabulary. We generated a corpus of 100 1000-word documents from a three-level hierarchy with a vocabulary of 25 terms. In this corpus, topics on the vocabulary can be viewed as bars on a $5 \times 5$ grid. The root topic places its probability mass on the bottom bar. On the second level, one topic is identified with the leftmost bar, while the rightmost bar represents a second topic. The leftmost topic has two subtopics while the rightmost topic has one subtopic. Figure 2a illustrates six documents sampled from this model. Figure 2b illustrates the recovered hierarchy using the Gibbs sampling algorithm described in Section 4.

In estimating hierarchy structures, hypothesis testing approaches to model selection are impractical since they do not provide a viable method of searching over the large space of trees. To compare the CRP method on LDA models with a standard approach, we implemented the simpler, flat model described at the end of Section 3. We generated 210 corpora of 100 1000-word documents each from an LDA model with $K \in \{5, \ldots, 25\}$, $L = 5$, a vocabulary size of 100, and randomly generated mixture components from a symmetric Dirichlet ($\eta = 0.1$). For comparison with the CRP prior, we used the approximate Bayes factors method of model selection [13], where one chooses the model that maximizes $p(\text{data} \mid K)p(K)$ for various $K$ and an appropriate prior. With the LDA model, the Bayes factors method is much slower than the CRP as it involves multiple runs of a Gibbs sampler with speed comparable to a single run of the CRP sampler. Furthermore, with the Bayes factors method one must choose an appropriate range of $K$. With the CRP prior, the only free parameter is $\gamma$ (we used $\gamma = 1.0$). As shown in Figure 4, the CRP prior was more effective than Bayes factors in this setting. We should note that both the CRP and Bayes factors are somewhat sensitive to the choice $\eta$, the hyperparameter to the prior on the topics. However, in simulated data, this hyperparameter was known and thus we can provide a fair comparison.

In a similar experiment, we generated 50 corpora each from five different hierarchies using

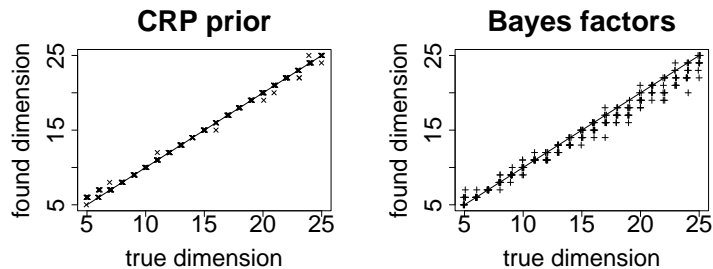

Figure 4: (Left) The average dimension found by a CRP prior plotted against the true dimension on simulated data (the true value is jiggled to see overlapping points). For each dimension, we generated ten corpora with a vocabulary size of 100. Each corpus contains 100 documents of 1000 words. (Right) Results of model selection with Bayes factors.

an hLDA model and the same symmetric Dirichlet prior on topics. Each corpus has 100 documents of 1000 words from a vocabulary of 100 terms. Figure 3 reports the results of sampling from the resulting posterior on trees with the Gibbs sampler from Section 4. In all cases, we recover the correct structure more than any other and we usually recover a structure within one leaf of the correct structure. In all experiments, no predicted structure deviated by more than three nodes from the correct structure.

Lastly, to demonstrate its applicability to real data, we applied the hLDA model to a text data set. Using 1717 NIPS abstracts from 1987–1999 [14] with 208,896 words and a vocabulary of 1600 terms, we estimated a three level hierarchy as illustrated in Figure 5. The model has nicely captured the function words without using an auxiliary list, a nuisance that most practical applications of language models require. At the next level, it separated the words pertaining to neuroscience abstracts and machine learning abstracts. Finally, it delineated several important subtopics within the two fields. These results suggest that hLDA can be an effective tool in text applications.

## 6 Summary

We have presented the nested Chinese restaurant process, a distribution on hierarchical partitions. We have shown that this process can be used as a nonparametric prior for a hierarchical extension to the latent Dirichlet allocation model. The result is a flexible, general model for topic hierarchies that naturally accommodates growing data collections. We have presented a Gibbs sampling procedure for this model which provides a simple method for simultaneously exploring the spaces of trees and topics.

Our model has two natural extensions. First, we have restricted ourselves to hierarchies of fixed depth $L$ for simplicity, but it is straightforward to consider a model in which $L$ can vary from document to document. Each document is still a mixture of topics along a path in a hierarchy, but different documents can express paths of different lengths as they represent varying levels of specialization. Second, although in our current model a document is associated with a single path, it is also natural to consider models in which documents are allowed to mix over paths. This would be a natural way to take advantage of syntactic structures such as paragraphs and sentences within a document.

## Acknowledgements

We wish to acknowledge support from the DARPA CALO program, Microsoft Corporation, and NTT Communication Science Laboratories.

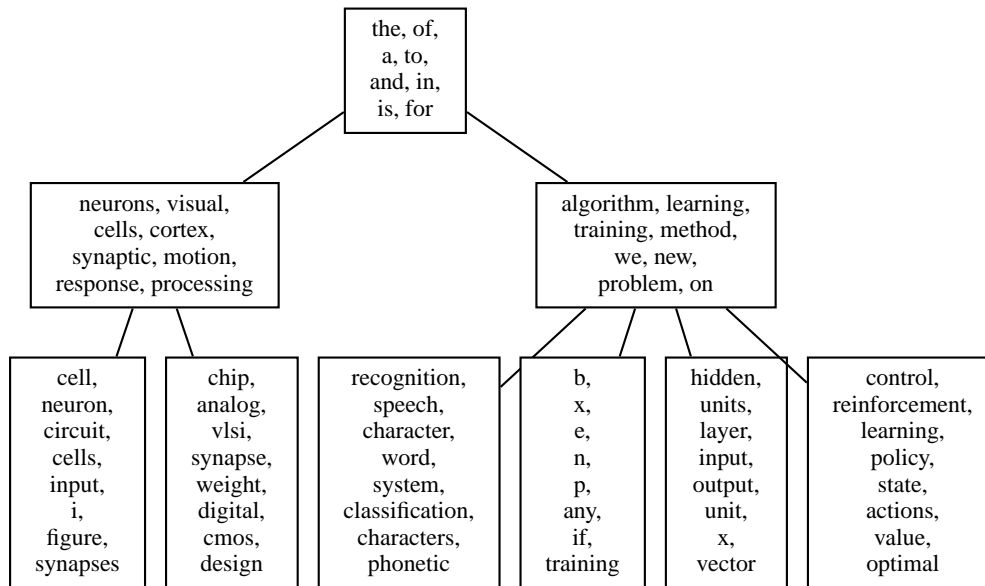

Figure 5: A topic hierarchy estimated from 1717 abstracts from NIPS01 through NIPS12. Each node contains the top eight words from its corresponding topic distribution.

## Footnotes

[1] The terminology was inspired by the Chinese restaurants in San Francisco which seem to have an infinite seating capacity. It was coined by Jim Pitman and Lester Dubins in the early eighties [1].

## References

[1] D. Aldous. Exchangeability and related topics. In *École d'été de probabilités de Saint-Flour, XIII—1983*, pages 1–198. Springer, Berlin, 1985.

[2] E. Segal, D. Koller, and D. Ormoneit. Probabilistic abstraction hierarchies. In *Advances in Neural Information Processing Systems 14*.

[3] T. Hofmann. The cluster-abstraction model: Unsupervised learning of topic hierarchies from text data. In *IJCAI*, pages 682–687, 1999.

[4] T. Ferguson. A Bayesian analysis of some nonparametric problems. *The Annals of Statistics*, 1:209–230, 1973.

[5] J. Pitman. *Combinatorial Stochastic Processes*. Notes for St. Flour Summer School. 2002.

[6] J. Ishwaran and L. James. Generalized weighted Chinese restaurant processes for species sampling mixture models. *Statistica Sinica*, 13:1211–1235, 2003.

[7] R. Neal. Markov chain sampling methods for Dirichlet process mixture models. *Journal of Computational and Graphical Statistics*, 9(2):249–265, June 2000.

[8] M. West, P. Muller, and M. Escobar. Hierarchical priors and mixture models, with application in regression and density estimation. In *Aspects of Uncertainty*. John Wiley.

[9] M. Beal, Z. Ghahramani, and C. Rasmussen. The infinite hidden Markov model. In *Advances in Neural Information Processing Systems 14*.

[10] C. Rasmussen and Z. Ghahramani. Infinite mixtures of Gaussian process experts. In *Advances in Neural Information Processing Systems 14*.

[11] D. Blei, A. Ng, and M. Jordan. Latent Dirichlet allocation. *Journal of Machine Learning Research*, 3:993–1022, January 2003.

[12] T. Griffiths and M. Steyvers. A probabilistic approach to semantic representation. In *Proceedings of the 24th Annual Conference of the Cognitive Science Society*, 2002.

[13] R. Kass and A. Raftery. Bayes factors. *Journal of the American Statistical Association*, 90(430):773–795, 1995.

[14] S. Roweis. NIPS abstracts, 1987–1999. http://www.cs.toronto.edu/ roweis/data.html.
